# Trajectory-Based Short-Sighted Probabilistic Planning

**Felipe W. Trevizan**
Machine Learning Department

**Manuela M. Veloso**
Computer Science Department

Carnegie Mellon University - Pittsburgh, PA
{fwt,mmv}@cs.cmu.edu

## Abstract

Probabilistic planning captures the uncertainty of plan execution by probabilistically modeling the effects of actions in the environment, and therefore the probability of reaching different states from a given state and action. In order to compute a solution for a probabilistic planning problem, planners need to manage the uncertainty associated with the different paths from the initial state to a goal state. Several approaches to manage uncertainty were proposed, e.g., consider all paths at once, perform determinization of actions, and sampling. In this paper, we introduce trajectory-based short-sighted Stochastic Shortest Path Problems (SSPs), a novel approach to manage uncertainty for probabilistic planning problems in which states reachable with low probability are substituted by artificial goals that heuristically estimate their cost to reach a goal state. We also extend the theoretical results of Short-Sighted Probabilistic Planner (SSiPP) [1] by proving that SSiPP always finishes and is asymptotically optimal under sufficient conditions on the structure of short-sighted SSPs. We empirically compare SSiPP using trajectory-based short-sighted SSPs with the winners of the previous probabilistic planning competitions and other state-of-the-art planners in the triangle tireworld problems. Trajectory-based SSiPP outperforms all the competitors and is the only planner able to scale up to problem number 60, a problem in which the optimal solution contains approximately $10^{70}$ states.

## 1 Introduction

The uncertainty of plan execution can be modeled by using probabilistic effects in actions, and therefore the probability of reaching different states from a given state and action. This search space, defined by the probabilistic paths from the initial state to a goal state, challenges the scalability of planners. Planners manage the uncertainty by choosing a search strategy to explore the space. In this work, we present a novel approach to manage uncertainty for probabilistic planning problems that improves its scalability while still being optimal.

One approach to manage uncertainty while searching for the solution of probabilistic planning problems is to consider the complete search space at once. Examples of such algorithms are value iteration and policy iteration [2]. Planners based on these algorithms return a closed policy, i.e., a universal mapping function from every state to the optimal action that leads to a goal state. Assuming the model correctly captures the cost and uncertainty of the actions in the environment, closed policies are extremely powerful as their execution never "fails," and the planner does not need to be re-invoked ever. Unfortunately the computation of such policies is prohibitive in complexity as problems scale up. Value iteration based probabilistic planners can be improved by combining asynchronous updates and heuristic search [3–7]. Although these techniques allow planners to compute compact policies, in the worst case, these policies are still linear in the size of the state space, which itself can be exponential in the size of the state or goals.

Another approach to manage uncertainty is basically to ignore uncertainty during planning, i.e., to approximate the probabilistic actions as deterministic actions. Examples of replanners based on determinization are FF-Replan [8], the winner of the first International Probabilistic Planning Competition (IPPC) [9], Robust FF [10], the winner of the third IPPC [11] and FF-Hindsight [12, 13]. Despite the major success of determinization, this simplification in the action space results in algorithms oblivious to probabilities and dead-ends, leading to poor performance in specific problems, e.g., the triangle tireworld [14].

Besides the action space simplification, uncertainty management can be performed by simplifying the problem horizon, i.e., look-ahead search [15]. Based on sampling, the Upper Confidence bound for Trees (UCT) algorithm [16] approximates the look-ahead search by focusing the search in the most promising nodes.

The state space can also be simplified to manage uncertainty in probabilistic planning. One example of such approach is Envelope Propagation (EP) [17]. EP computes an initial partial policy $\pi$ and then prunes all the states not considered by $\pi$. The pruned states are represented by a special meta state. Then EP iteratively improves its approximation of the state space. Previously, we introduced short-sighted planning [1], a new approach to manage uncertainty in planning problems: given a state $s$, only the uncertainty structure of the problem in the neighborhood of $s$ is taken into account and the remaining states are approximated by artificial goals that heuristically estimate their cost to reach a goal state.

In this paper, we introduced trajectory-based short-sighted Stochastic Shortest Path Problems (SSPs), a novel model to manage uncertainty in probabilistic planning problems. Trajectory-based short-sighted SSPs manage uncertainty by pruning the state space based on the most likely trajectory between states and defining artificial goal states that guide the solution towards the original goal. We also contribute by defining a class of short-sighted models and proving that the Short-Sighted Probabilistic Planner (SSiPP) [1] always terminates and is asymptotically optimal for models in this class of short-sighted models.

The remainder of this paper is organized as follows: Section 2 introduces the basic concepts and notation. Section 3 defines formally trajectory-based short-sighted SSPs. Section 4 presents our new theoretical results for SSiPP. Section 5 empirically evaluates SSiPP using trajectory-based short-sighted SSPs with the winners of the previous IPPCs and other state-of-the-art planner. Section 6 concludes the paper.

## 2 Background

A Stochastic Shortest Path Problem (SSP) is defined by the tuple $\mathbb{S} = \langle \mathsf{S}, s_0, \mathsf{G}, \mathsf{A}, P, C \rangle$, in which [1, 18]: $\mathsf{S}$ is the finite set of state; $s_0 \in \mathsf{S}$ is the initial state; $\mathsf{G} \subseteq \mathsf{S}$ is the set of goal states; $\mathsf{A}$ is the finite set of actions; $P(s'|s, a)$ represents the probability that $s' \in \mathsf{S}$ is reached after applying action $a \in \mathsf{A}$ in state $s \in \mathsf{S}$; $C(s, a, s') \in (0, +\infty)$ is the cost incurred when state $s'$ is reached after applying action $a$ in state $s$ and this function is required to be defined for all $s \in \mathsf{S}, a \in \mathsf{A}, s' \in \mathsf{S}$ such that $P(s'|s, a) > 0$.

A solution to an SSP is a policy $\pi$, i.e., a mapping from $\mathsf{S}$ to $\mathsf{A}$. If $\pi$ is defined over the entire space $\mathsf{S}$, then $\pi$ is a closed policy. A policy $\pi$ defined only for the states reachable from $s_0$ when following $\pi$ is a closed policy w.r.t. $s_0$ and $\mathsf{S}(\pi, s_0)$ denotes this set of reachable states. For instance, in the SSP depicted in Figure 1(a), the policy $\pi_0 = \{(s_0, a_0), (s'_1, a_0), (s'_2, a_0), (s'_3, a_0)\}$ is a closed policy w.r.t. $s_0$ and $\mathsf{S}(\pi_0, s_0) = \{s_0, s'_1, s'_2, s'_3, s_\mathsf{G}\}$.

Given a policy $\pi$, we define trajectory as a sequence $\mathcal{T}_\pi = \langle s_{(0)}, \dots, s_{(k)} \rangle$ such that, for all $i \in \{0, \cdots, k-1\}$, $\pi(s_{(i)})$ is defined and $P(s_{(i+1)}|s_{(i)}, \pi(s_{(i)})) > 0$. The probability of a trajectory $\mathcal{T}_\pi$ is defined as $P(\mathcal{T}_\pi) = \prod_{i=0}^{i<|\mathcal{T}_\pi|} P(s_{(i+1)}|s_{(i)}, \pi(s_{(i)}))$ and maximum probability of a trajectory between two states $P_{\max}(s, s')$ is defined as $\max_\pi P(\mathcal{T}_\pi = \langle s, \dots, s' \rangle)$.

An optimal policy $\pi^*$ for an SSP is any policy that always reaches a goal state when followed from $s_0$ and also minimizes the expected cost of $\mathcal{T}_{\pi^*}$. For a given SSP, $\pi^*$ might not be unique, however the optimal value function $V^*$, i.e., the mapping from states to the minimum expected cost to reach a goal state, is unique. $V^*$ is the fixed point of the set of equations defined by (1) for all $s \in \mathsf{S} \setminus \mathsf{G}$ and $V^*(s) = 0$ for all $s \in \mathsf{G}$. Notice that under the optimality criterion given by (1), SSPs are

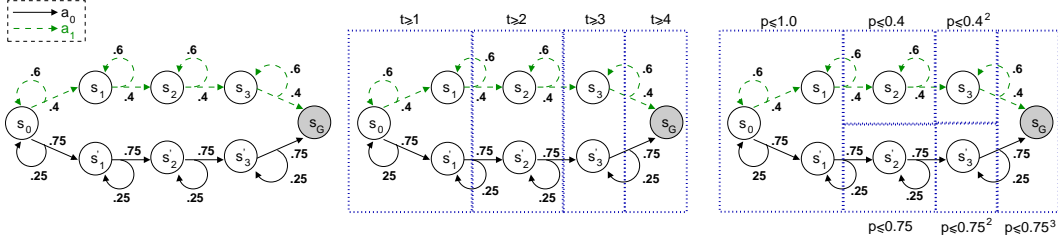

Figure 1: (a) Example of an SSP. The initial state is $s_0$, the goal state is $s_\mathbf{G}$, $C(s, a, s') = 1$, $\forall s \in \mathbf{S}$, $a \in \mathbf{A}$ and $s' \in \mathbf{S}$. (b) State-space partition of (a) according to the depth-based short-sighted SSPs: $\mathbf{G}_{s_0,t}$ contains all the states in dotted regions which their conditions hold for the given value of $t$. (c) State-space partition of (a) according to the trajectory-based short-sighted SSPs: $\mathbf{G}_{s_0,\rho}$ contains all the states in dotted regions which their conditions hold for the given value of $\rho$.

more general than Markov Decision Processes (MDPs) [19], therefore all the work presented here is directly applicable to MDPs.

$$V^*(s) = \min_{a \in \mathbf{A}} \sum_{s' \in \mathbf{S}} \left[ C(s, a, s') + P(s'|s, a)V^*(s') \right] \tag{1}$$

**Definition 1** (reachability assumption). *An SSP satisfies the reachability assumption if, for all $s \in \mathbf{S}$, there exists $s_G \in \mathbf{G}$ such that $P_{\max}(s, s_G) > 0$.*

Given an SSP $\mathbb{S}$, if a goal state can be reached with positive probability from every state $s \in \mathbf{S}$, then the reachability assumption (Definition 1) holds for $\mathbb{S}$ and $0 \le V^*(s) < \infty$ [19]. Once $V^*$ is known, any optimal policy $\pi^*$ can be extracted from $V^*$ by substituting the operator $\min$ by $\operatorname{argmin}$ in equation (1).

A possible approach to compute $V^*$ is the value iteration algorithm: define $V^{i+1}(s)$ as in (1) with $V^i$ in the right hand side instead of $V^*$ and the sequence $\langle V^0, V^1, \ldots, V^k \rangle$ converges to $V^*$ as $k \to \infty$ [19]. The process of computing $V^{i+1}$ from $V^i$ is known as Bellman update and $V^0(s)$ can be initialized with an admissible heuristic $H(s)$, i.e., a lower bound for $V^*$. In practice we are interested in reaching $\epsilon$-convergence, that is, given $\epsilon$, find $V$ such that $\max_s |V(s) - \min_a \sum_{s'} [C(s, a, s') + P(s'|s, a)V(s')]| \le \epsilon$. The following well-known result is necessary in most of our proofs [2, Assumption 2.2 and Lemma 2.1]:

**Theorem 1.** *Given an SSP $\mathbb{S}$, if the reachability assumption holds for $\mathbb{S}$, then the admissibility and monotonicity of $V$ are preserved through the Bellman updates.*

## 3 Trajectory-Based Short-Sighted Stochastic SSPs

Short-sighted Stochastic Path Problems (short-sighted SSPs) [1] are a special case of SSPs in which the original problem is transformed into a smaller one by: (i) pruning the state space; and (ii) adding artificial goal states to heuristically guide the search towards the goals of the original problem. Depth-based short-sighted SSPs are defined based on the action-distance between states [1]:

**Definition 2** (action-distance). *The non-symmetric action-distance $\delta(s, s')$ between two states $s$ and $s'$ is $\operatorname{argmin}_k \{ \mathcal{T}_\pi = \langle s, s_{(1)}, \ldots, s_{(k-1)}, s' \rangle | \exists \pi \text{ and } \mathcal{T}_\pi \text{ is a trajectory} \}$.*

**Definition 3** (Depth-Based Short-Sighted SSP). *Given an SSP $\mathbb{S} = \langle \mathbf{S}, s_0, \mathbf{G}, \mathbf{A}, P, C \rangle$, a state $s \in \mathbf{S}$, $t > 0$ and a heuristic $H$, the $(s, t)$-depth-based short-sighted SSP $\mathbb{S}_{s,t} = \langle \mathbf{S}_{s,t}, s, \mathbf{G}_{s,t}, \mathbf{A}, P, C_{s,t} \rangle$ associated with $\mathbb{S}$ is defined as:*

- $\mathbf{S}_{s,t} = \{ s' \in \mathbf{S} | \delta(s, s') \le t \}$;
- $\mathbf{G}_{s,t} = \{ s' \in \mathbf{S} | \delta(s, s') = t \} \cup (\mathbf{G} \cap \mathbf{S}_{s,t})$;
- $C_{s,t}(s', a, s'') = \begin{cases} C(s', a, s'') + H(s'') & \text{if } s'' \in \mathbf{G}_{s,t} \\ C(s', a, s'') & \text{otherwise} \end{cases}$, $\quad \forall s' \in \mathbf{S}_{s,t}, a \in \mathbf{A}, s'' \in \mathbf{S}_{s,t}$

Figure 1(b) shows, for different values of $t$, $\mathbf{S}_{s_0,t}$ for the SSP in Figure 1(a); for instance, if $t = 2$ then $\mathbf{S}_{s_0,2} = \{s_0, s_1, s_1', s_2, s_2'\}$ and $\mathbf{G}_{s_0,2} = \{s_2, s_2'\}$. In the example shown in Figure 1(b), we can

see that generation of $\mathbf{S}_{s_0,t}$ is independent of the trajectories probabilities: for $t = 2$, $s_2 \in \mathbf{S}_{s_0,2}$ and $s_3' \notin \mathbf{S}_{s_0,2}$, however $P_{\max}(s_0, s_2) = 0.16 < P_{\max}(s_0, s_3') = 0.75^3 \approx 0.42$.

**Definition 4** (Trajectory-Based Short-Sighted SSP). *Given an SSP $\mathbb{S} = \langle \mathbf{S}, s_0, \mathbf{G}, \mathbf{A}, P, C \rangle$, a state $s \in \mathbf{S}$, $\rho \in [0,1]$ and a heuristic $H$, the $(s,\rho)$-trajectory-based short-sighted SSP $\mathbb{S}_{s,\rho} = \langle \mathbf{S}_{s,\rho}, s, \mathbf{G}_{s,\rho}, \mathbf{A}, P, C_{s,\rho} \rangle$ associated with $\mathbb{S}$ is defined as:*

- $\mathbf{S}_{s,\rho} = \{ s' \in \mathbf{S} | \exists \hat{s} \in \mathbf{S} \text{ and } a \in \mathbf{A} \text{ s.t. } P_{\max}(s, \hat{s}) \geq \rho \text{ and } P(s'|\hat{s}, a) > 0 \}$;

- $\mathbf{G}_{s,\rho} = (\mathbf{G} \cap \mathbf{S}_{s,\rho}) \cup (\mathbf{S}_{s,\rho} \cap \{ s' \in \mathbf{S} | P_{\max}(s, s') < \rho \})$;

- $C_{s,\rho}(s', a, s'') = \begin{cases} C(s', a, s'') + H(s'') & \text{if } s'' \in \mathbf{G}_{s,\rho} \\ C(s', a, s'') & \text{otherwise} \end{cases}$, $\quad \forall s' \in \mathbf{S}_{s,\rho}, a \in \mathbf{A}, s'' \in \mathbf{S}_{s,\rho}$

*For simplicity, when $H$ is not clear by context nor explicit, then $H(s) = 0$ for all $s \in \mathbf{S}$.*

Our novel model, the trajectory-based short-sighted SSPs (Definition 4), addresses the issue of states with low trajectory probability by explicitly defining its state space $\mathbf{S}_{s,\rho}$ based on maximum probability of a trajectory between $s$ and the candidate states $s'$ ($P_{\max}(s, s')$). Figure 1(c) shows, for all values of $\rho \in [0,1]$, the trajectory-based $\mathbf{S}_{s_0,\rho}$ for the SSP in Figure 1(a): for instance, if $\rho = 0.75^3$ then $\mathbf{S}_{s_0,0.75^3} = \{s_0, s_1, s_1', s_2', s_3', s_G\}$ and $\mathbf{G}_{s_0,0.75} = \{s_1, s_G\}$. This example shows how trajectory-based short-sighted SSP can manage uncertainty efficiently: for $\rho = 0.75^3$, $|\mathbf{S}_{s_0,\rho}| = 6$ and the goal of the original SSP $s_G$ is already included in $\mathbf{S}_{s_0,\rho}$ while, for the depth-based short-sighted SSPs, $s_G \in \mathbf{S}_{s_0,t}$ only for $t \geq 4$ case in which $|\mathbf{S}_{s_0,t}| = |\mathbf{S}| = 8$.

Notice that the definition of $\mathbf{S}_{s,\rho}$ cannot be simplified to $\{ \hat{s} \in \mathbf{S} | P_{\max}(s, \hat{s}) \geq \rho \}$ since not all the resulting states of actions would be included in $\mathbf{S}_{s,\rho}$. For example, consider $\mathbf{S} = \{s, s', s''\}$, $P(s'|s, a) = 0.9$ and $P(s''|s, a) = 0.1$, then for $\rho \in (0.1, 1]$, $\{ \hat{s} \in \mathbf{S} | P_{\max}(s, \hat{s}) \geq \rho \} = \{s, s'\}$, generating an invalid SSP since not all the resulting states of $a$ would be contained in the model.

## 4 Short-Sighted Probabilistic Planner

The Short-Sighted Probabilistic Planner (SSiPP) is an algorithm that solves SSPs based on short-sighted SSPs [1]. SSiPP is reviewed in Algorithm 1 and consists of iteratively generating and solving short-sighted SSPs of the given SSP. Due to the reduced size of the short-sighted problems, SSiPP solves each of them by computing a closed policy w.r.t. their initial state. Therefore, we obtain a "fail-proof" solution for each short-sighted SSP, thus if this solution is directly executed in the environment, then replanning is not needed until an artificial goal is reached. Alternatively, an anytime behavior is obtained if the execution of the computed closed policy for the short-sighted SSP is simulated (Algorithm 1 line 4) until an artificial goal $s_a$ is reached and this procedure is repeated, starting $s_a$, until convergence or an interruption.

In [1], we proved that SSiPP always terminates and is asymptotically optimal for depth-based short-sighted SSPs. We generalize the results regarding SSiPP by: (i) providing the sufficient conditions for the generation of short-sighted problems (Algorithm 1, line 1) in Definition 5; and (ii) proving that SSiPP always terminates (Theorem 3) and is asymptotically optimal (Corollary 4) when the short-sighted SSP generator respects Definition 5. Notice that, by definition, both depth-based and trajectory-based short-sighted SSPs meet the sufficient conditions presented on Definition 5.

**Definition 5.** *Given an SSP $\langle \mathbf{S}, s_0, \mathbf{G}, \mathbf{A}, P, C \rangle$, the sufficient conditions on the short-sighted SSPs $\langle \mathbf{S}', \hat{s}, \mathbf{G}', \mathbf{A}, P', C' \rangle$ returned by the generator in Algorithm 1 line 1 are:*

1. *$\mathbf{G} \cap \mathbf{S}' \subseteq \mathbf{G}'$;*

2. *$\hat{s} \notin \mathbf{G} \rightarrow \hat{s} \notin \mathbf{G}'$; and*

3. *for all $s \in \mathbf{S}, a \in \mathbf{A}$ and $s' \in \mathbf{S}' \setminus \mathbf{G}'$, if $P(s|s', a) > 0$ then $s \in \mathbf{S}'$ and $P'(s|s', a) = P(s|s', a)$.*

**Lemma 2.** *SSiPP performs Bellman updates on the original SSP $\mathbb{S}$.*

SSiPP(SSP $\mathbb{S} = \langle \mathbf{S}, s_0, \mathbf{G}, \mathbf{A}, P, C \rangle$, $H$ a heuristic for $V^*$ and *params* the parameters to generate short-sighted SSPs)

**begin**

    $\underline{V} \leftarrow$ Value function for $\mathbf{S}$ initialized by $H$

    $s \leftarrow s_0$

    **while** $s \notin \mathbf{G}$ **do**

1         $\langle \mathbf{S}', s, \mathbf{G}', \mathbf{A}, P, C' \rangle \leftarrow$ GENERATE-SHORT-SIGHTED-SSP($\mathbb{S}, s, \underline{V}$, *params*)

        $(\hat{\pi}^*, \hat{V}^*) \leftarrow$ OPTIMAL-SSP-SOLVER($\langle \mathbf{S}', s, \mathbf{G}', \mathbf{A}, P, C' \rangle, \underline{V}$)

2         **forall** $s' \in \mathbf{S}'(\hat{\pi}^*, s)$ **do**

           $\underline{V}(s') \leftarrow \hat{V}^*(s')$

3         **while** $s \notin \mathbf{G}'$ **do**

4            $s \leftarrow$ execute-action($\hat{\pi}^*(s)$)

    **return** $\underline{V}$

**end**

**Algorithm 1**: SSiPP algorithm [1]. GENERATE-SHORT-SIGHTED-SSP represents a procedure to generate short-sighted SSPs, either depth-based or trajectory-based. In the former case *params = t* and *params = $\rho$* for the latter. OPTIMAL-SSP-SOLVER returns an optimal policy $\pi^*$ w.r.t. $s_0$ for $\mathbb{S}$ and $V^*$ associated to $\pi^*$, i.e., $V^*$ needs to be defined only for $s \in \mathbf{S}(\pi^*, s_0)$.

*Proof.* In order to show that SSiPP performs Bellman updates implicitly, consider the loop in line 2 of Algorithm 1. Since OPTIMAL-SOLVER computes $\hat{V}^*$, by definition of short-sighted SSP: (i) $\hat{V}^*(s_{\mathbf{G}})$ equals $\underline{V}(s_{\mathbf{G}})$ for all $s_{\mathbf{G}} \in \mathbf{G}'$, therefore the value of $\underline{V}(s_{\mathbf{G}})$ remains the same; and (ii) $\min_{a \in \mathbf{A}} \sum_{s' \in \mathbf{S}} [C(s, a, s') + P(s'|s, a)\underline{V}(s')] \leq \hat{V}^*(s)$ for $s \in \mathbf{S}' \setminus \mathbf{G}'$, i.e., the assignment $\underline{V}(s) \leftarrow \hat{V}^*$ is equivalent to at least one Bellman update on $\underline{V}(s)$, because $\underline{V}$ is a lower bound on $\hat{V}^*$ and Theorem 1. Because $s \notin \mathbf{G}'$ and Definition 5, $\min_{a \in \mathbf{A}} \left[ \sum_{s' \in \mathbf{S}} C(s, a, s') + P(s'|s, a)\underline{V}(s') \right] \leq \hat{V}^*(s)$ is equivalent to the one Bellman update in the original SSP $\mathbb{S}$. $\square$

**Theorem 3.** *Given an SSP $\mathbb{S} = \langle \mathbf{S}, s_0, \mathbf{G}, \mathbf{A}, P, C \rangle$ such that the reachability assumption holds, an admissible heuristic $H$ and a short-sighted problem generator that respects Definition 5, then SSiPP always terminates.*

*Proof.* Since OPTIMAL-SOLVER always finishes and the short-sighted SSP is an SSP by definition, then a goal state $s_G$ of the short-sighted SSP is always reached, therefore the loop in line 3 of Algorithm 1 always finishes. If $s_G \in \mathbf{G}$, then SSiPP terminates in this iteration. Otherwise, $s_G$ is an artificial goal and $s_G \neq s$ (Definition 5), i.e., $s_G$ differs from the state $s$ used as initial state for the short-sighted SSP generation. Thus another iteration of SSiPP is performed using $s_G$ as $s$. Suppose, for contradiction purpose, that every goal state reached during SSiPP execution is an artificial goal, i.e., SSiPP does not terminate. Then infinitely many short-sighted SSPs are solved. Since $\mathbf{S}$ is finite, then there exists $s \in \mathbf{S}$ that is updated infinitely often, therefore $\underline{V}(s) \to \infty$. However, $V^*(s) < \infty$ by the reachability assumption. Since SSiPP performs Bellman updates (Lemma 2) then $\underline{V}(s) \leq V^*(s)$ by monotonicity of Bellman updates (Theorem 1) and admissibility of $H$, a contradiction. Thus every execution of SSiPP reaches a goal state $s'_G \in \mathbf{G}$ and therefore terminates. $\square$

**Corollary 4.** *Under the same assumptions of Theorem 3, the sequence $\langle \underline{V}_0, \underline{V}_1, \cdots, \underline{V}_t \rangle$, where $\underline{V}_0 = H$ and $\underline{V}_t = SSiPP(\mathbb{S}, t, \underline{V}_{t-1})$, converges to $V^*$ as $t \to \infty$ for all $s \in \mathbf{S}(\pi^*, s_0)$.*

*Proof.* Let $\mathbf{S}^* \subseteq \mathbf{S}$ be the set of states being visited infinitely many times. Clearly, $\mathbf{S}(\pi^*, s_0) \subseteq \mathbf{S}^*$ since a partial policy cannot be executed ad infinitum without reaching a state in which it is not defined. Since SSiPP performs Bellman updates in the original SSP space (Lemma 2) and every execution of SSiPP terminates (Theorem 3), then we can view the sequence of lower bounds $\langle \underline{V}_0, \underline{V}_1, \cdots, \underline{V}_t \rangle$ generated by SSiPP as asynchronous value iteration. The convergence of $\underline{V}_{t-1}(s)$ to $V^*(s)$ as $t \to \infty$ for all $s \in \mathbf{S}(\pi^*, s_0) \subseteq \mathbf{S}^*$ follows by [2, Proposition 2.2, p. 27] and guarantees the convergence of SSiPP. $\square$

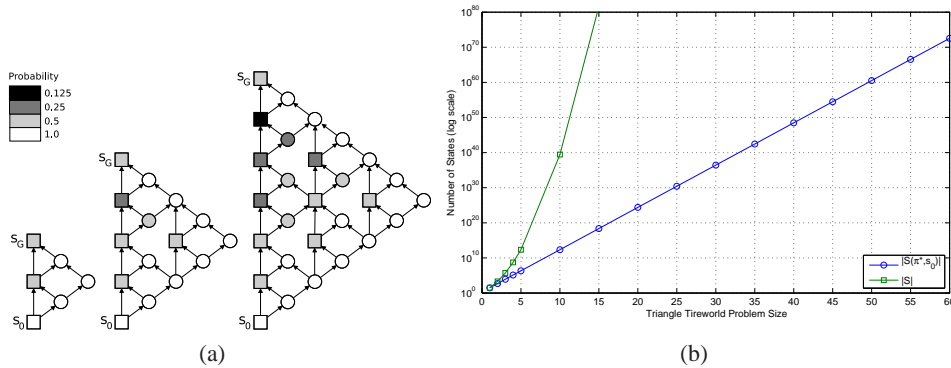

(a)                                                                    (b)

Figure 2: (a) Map of the triangle tireworld for the sizes 1, 2 and 3. Circles (squares) represent locations in which there is one (no) spare tire. The shades of gray represent, for each location $l$, $\max_\pi P$(car reaches $l$ and the tire is not flat when following the policy $\pi$ from $s_0$). (b) Log-lin plot of the state space size ($|\mathbf{S}|$) and the size of the states reachable from $s_0$ when following the optimal policy $\pi^*$ ($|\mathbf{S}(\pi^*, s_0)|$) versus the number of the triangle tireworld problem.

## 5  Experiments

We present two sets of experiments using the triangle tireworld problems [9, 11, 20], a series of probabilistic interesting problems [14] in which a car has to travel between locations in order to reach a goal location from its initial location. The roads are represented as directed graph in a shape of a triangle and, every time the car moves between locations, a flat tire happens with probability $0.5$. Some locations have a spare tire and in these locations the car can deterministically replace its flat tire by new one. When the car has a flat tire, it cannot change its location, therefore the car can get stuck in locations that do not have a spare tire (dead-ends). Figure 2(a) depicts the map of the triangle tireworld problems 1, 2 and 3 and Figure 2(b) shows the size of $\mathbf{S}$ and $\mathbf{S}(\pi^*, s_0)$ for problems up to size 60. For example, the problem number 3 has 28 locations, i.e., 28 nodes in the corresponding graph on Figure 2(a), its state space has 19562 states and its optimal policy reaches 8190 states.

Every triangle tireworld problem is a probabilistic interesting problem [14]: there is only one policy that reaches the goal with probability 1 and all the other policies have probability at most $0.5$ of reaching the goal. Also, the solution based on the shortest path has probability $0.5^{2n-1}$ of reaching the goal, where $n$ is the problem number. This property is illustrated by the shades of gray in Figure 2(a) that represents, for each location $l$, $\max_\pi P$(car reaches $l$ and the tire is not flat when following the policy $\pi$ from $s_0$).

For the experiments in this section, we use the zero-heuristic for all the planners, i.e., $\underline{V}(s) = 0$ for all $s \in \mathbf{S}$ and LRTDP [4] as OPTIMAL-SOLVER for SSiPP. For all planners, the parameter $\epsilon$ (for $\epsilon$-convergence) is set to $10^{-4}$. For UCT, we disabled the random rollouts because the probability of any policy other than the optimal policy to reach a dead-end is at least $0.5$ therefore, with high-probability, UCT would assign $\infty$ (cost of a dead-end) as the cost of all the states including the initial state.

The experiments are conducted in a Linux machine with 4 cores running at 3.07GHz using MDP-SIM [9] as environment simulator. The following terminology is used for describing the experiments: *round*, the computation for a solution for the given SSP; and *run*, a set of rounds in which learning is allowed between rounds, i.e., the knowledge obtained from one round can be used to solve subsequent rounds. The solution computed during one round is simulated by MDPSIM in a client-server loop: MDPSIM sends a state $s$ and requests an action from the planner, then the planner replies by sending the action $a$ to be executed in $s$. The evaluation is done by the number of rounds simulated by MDPSIM that reached a goal state. The maximum number of actions allowed per round is 2000 and rounds that exceed this limit are stopped by MDPSIM and declared as failure, i.e., goal not reached.

| | Triangle Tireworld Problem Number | | | | | | | | | | | |
|---|---|---|---|---|---|---|---|---|---|---|---|---|
| Planner | 5 | 10 | 15 | 20 | 25 | 30 | 35 | 40 | 45 | 50 | 55 | 60 |
| SSiPP depth=8 | **50.0** | 40.7 | 41.2 | 40.8 | 41.1 | 41.0 | 40.9 | 40.0 | 40.6 | 40.8 | 40.3 | 40.4 |
| UCT | **50.0** | **50.0** | **50.0** | **50.0** | **50.0** | 43.1 | 15.7 | 12.1 | 8.2 | 6.8 | 5.0 | 4.0 |
| SSiPP trajectory | **50.0** | **50.0** | **50.0** | **50.0** | **50.0** | **50.0** | **50.0** | **50.0** | **50.0** | **50.0** | **50.0** | **50.0** |

Table 1: Number of rounds solved out of 50 for experiment in Section 5.1. Results are averaged over 10 runs and the 95% confidence interval is always less than 1.0. In all the problems, SSiPP using trajectory-based short-sighted SSPs solves all the 50 round in all the 10 runs, therefore its 95% confidence interval is 0.0 for all the problems. Best results shown in bold font.

| | Triangle Tireworld Problem Number | | | | | | | | | | | |
|---|---|---|---|---|---|---|---|---|---|---|---|---|
| Planner | 5 | 10 | 15 | 20 | 25 | 30 | 35 | 40 | 45 | 50 | 55 | 60 |
| SSiPP depth=8 | **50.0** | 45.4 | 41.2 | 42.3 | 41.2 | 44.1 | 42.4 | 32.7 | 20.6 | 14.1 | 9.9 | 7.0 |
| LRTDP | **50.0** | 23.0 | 14.1 | 0.3 | - | - | - | - | - | - | - | - |
| UCT $(4, 100)$ | **50.0** | **50.0** | **50.0** | 48.8 | 24.0 | 12.3 | 6.5 | 4.0 | 2.5 | 1.3 | 1.0 | 0.7 |
| UCT $(8, 100)$ | **50.0** | **50.0** | **50.0** | 46.3 | 24.0 | 12.3 | 6.7 | 3.7 | 2.2 | 1.2 | 1.0 | 0.6 |
| UCT $(2, 100)$ | **50.0** | **50.0** | **50.0** | 49.5 | 23.2 | 12.0 | 7.5 | 3.5 | 2.2 | 1.2 | 1.0 | 0.6 |
| SSiPP $\rho = 1.0$ | **50.0** | 27.9 | 29.1 | 26.8 | 26.0 | 26.6 | 28.6 | 27.2 | 26.6 | 27.6 | 26.2 | 26.9 |
| SSiPP $\rho = 0.50$ | **50.0** | **50.0** | **50.0** | **50.0** | **50.0** | **50.0** | **50.0** | **50.0** | **50.0** | **50.0** | **50.0** | **50.0** |
| SSiPP $\rho = 0.25$ | **50.0** | **50.0** | **50.0** | **50.0** | 47.6 | 45.0 | 41.1 | 42.7 | 41.9 | 40.7 | 40.1 | 40.4 |
| SSiPP $\rho = 0.125$ | **50.0** | **50.0** | **50.0** | **50.0** | **50.0** | **50.0** | **50.0** | **50.0** | 49.8 | 37.4 | 26.4 | 18.9 |

Table 2: Number of rounds solved out of 50 for experiment in Section 5.2. Results are averaged over 10 runs and the 95% confidence interval is always less than 2.6. UCT $(c, w)$ represents UCT using $c$ as bias parameter and $w$ samples per decision. In all the problems, trajectory-based SSiPP for $\rho = 0.5$ solves all the 50 round in all the 10 runs, therefore its 95% confidence interval is 0.0 for all the problems. Best results shown in bold font.

## 5.1 Fixed number of search nodes per decision

In this experiment, we compare the performance of UCT, depth-based SSiPP, and trajectory-based SSiPP with respect to the number of nodes explored by depth-based SSiPP. Formally, to decide what action to apply in a given state $s$, each planner is allowed to use at most $B = |\mathbf{S}_{s,t}|$ search nodes, i.e., the size of the search space is bounded by the equivalent $(s, t)$-short-sighted SSP. We choose $t$ equals to 8 since it obtains the best performance in the triangle tireworld problems [1]. Given the search nodes budget $B$, for UCT we sample the environment until the search tree contains $B$ nodes; and for trajectory-based SSiPP we use $\rho = \mathrm{argmax}_\rho \{|\mathbf{S}_{s,\rho}|$ s.t. $B \geq |\mathbf{S}_{s,\rho}|\}$.

The methodology for this experiment is as follows: for each problem, 10 runs of 50 rounds are performed for each planner using the search nodes budget $B$. The results, averaged over the 10 runs, are presented in Table 1. We set as time and memory cut-off 8 hours and 8 Gb, respectively, and UCT for problems 35 to 60 was the only planner preempted by the time cut-off. Trace-based SSiPP outperforms both depth-based SSiPP and UCT, solving all the 50 rounds in all the 10 runs for all the problems.

## 5.2 Fixed maximum planning time

In this experiment, we compare planners by limiting the maximum planning time. The methodology used in this experiment is similar to the one in IPPC'04 and IPPC'06: for each problem, planners need to solve 1 run of 50 rounds in 20 minutes. For this experiment, the planners are allowed to perform internal simulations, for instance, a planner can spend 15 minutes solving rounds using internal simulations and then use the computed policy to solve the required 50 rounds through MDPSIM in the remaining 5 minutes. The memory cut-off is 3Gb.

For this experiment, we consider the following planners: depth-based SSiPP for $t = 8$ [1], trajectory-based SSiPP for $\rho \in \{1.0, 0.5, 0.25, 0.125\}$, LRTDP using 3-look-ahead [1] and 12 different parametrizations of UCT obtained by using the bias parameter $c \in \{1, 2, 4, 8\}$ and the number of samples per decision $w \in \{10, 100, 1000\}$. The winners of IPPC'04, IPPC'06 and IPPC'08 are

omitted since their performance on the triangle tireworld problems are strictly dominated by depth-base SSiPP for $t = 8$. Table 2 shows the results of this experiment and due to space limitations we show only the top 3 parametrizations of UCT: 1st ($c = 4, w = 100$); 2nd ($c = 8, w = 100$); and 3rd ($c = 2, w = 100$).

All the four parametrizations of trajectory-based SSiPP outperform the other planners for problems of size equal or greater than 45. Trajectory-based SSiPP using $\rho = 0.5$ is especially noteworthy because it achieves the perfect score in all problems, i.e., it reaches a goal state in all the 50 rounds in all the 10 runs for all the problems. The same happens for $\rho = 0.125$ and problems up to size 40. For larger problems, trajectory-based SSiPP using $\rho = 0.125$ reaches the 20 minutes time cut-off before solving 50 rounds, however all the solved rounds successfully reach the goal. This interesting behavior of trajectory-based SSiPP for the triangle tireworld can be explained by the following theorem:

**Theorem 5.** *For the triangle tireworld, trajectory-based SSiPP using an admissible heuristic never falls in a dead-end for $\rho \in (0.5^{i+1}, 0.5^i]$ for $i \in \{1, 3, 5, \dots\}$.*

*Proof Sketch.* The optimal policy for the triangle tireworld is to follow the longest path: move from the initial location $l_0$ to the goal location $l_G$ passing through location $l_c$, where $l_0$, $l_c$ and $l_G$ are the vertices of the triangle formed by the problem's map. The path from $l_c$ to $l_G$ is unique, i.e., there is only one applicable move-car action for all the locations in this path. Therefore all the decision making to find the optimal policy happens between the locations $l_0$ and $l_c$. Each location $l'$ in the path from $l_0$ to $l_c$ has either two or three applicable move-car actions and we refer to the set of locations $l'$ with three applicable move-car actions as **N**. Every location $l' \in$ **N** is reachable from $l_0$ by applying an even number of move-car actions (Figure 2(a)) and the three applicable move-car actions in $l'$ are: (i) the optimal action $a_c$, i.e., move the car towards $l_c$; (ii) the action $a_G$ that moves the car towards $l_G$; and (iii) the action $a_p$ that moves the car parallel to the shortest-path from $l_0$ to $l_G$. The location reached by $a_p$ does not have a spare tire, therefore $a_p$ is never selected by a greedy choice over any admissible heuristic since it reaches a dead-end with probability 0.5. The locations reached by applying either $a_c$ or $a_G$ have a spare tire and the greedy choice between them depends on the admissible heuristic used, thus $a_G$ might be selected instead of $a_c$. However, after applying $a_G$, only one move-car action $a$ is available and it reaches a location that does not have a spare tire. Therefore, the greedy choice between $a_c$ and $a_G$ considering two or more move-car actions is optimal under any admissible heuristic: every sequence of actions $\langle a_G, a, \dots \rangle$ reaches a dead-end with probability at least 0.5 and at least one sequence of actions starting with $a_c$ has probability 0 to reach a dead-end, e.g., the optimal solution.

Given $\rho$, we denote as $\mathbf{L}_{s,\rho}$ the set of all locations corresponding to states in $\mathbf{S}_{s,\rho}$ and as $l_s$ the location corresponding to the state $s$. Thus, $\mathbf{L}_{s,\rho}$ contains all the locations reachable from $l_s$ using up to $m = \lfloor \log_{0.5} \rho \rfloor + 1$ move-car actions. If $m$ is even and $l_s \in$ **N**, then every location in $\mathbf{L}_{s,\rho} \cap$ **N** represents a state either in $\mathbf{G}_{s,\rho}$ or at least two move-car actions away from any state in $\mathbf{G}_{s,\rho}$. Therefore the solution of the $(s, \rho)$-trajectory-based short-sighted SSP only chooses the action $a_c$ to move the car. Also, since $m$ is even, every state $s$ used by SSiPP for generating $(s, \rho)$-trajectory-based short-sighted SSPs has $l_s \in$ **N**. Therefore, for even values of $m$, i.e., for $\rho \in (0.5^{i+1}, 0.5^i]$ and $i \in \{1, 3, 5, \dots\}$, trajectory-based SSiPP always chooses the actions $a_c$ to move the car to $l_c$, thus avoiding the all dead-ends. $\square$

## 6 Conclusion

In this paper, we introduced trajectory-based short-sighted SSPs, a new model to manage uncertainty in probabilistic planning problems. This approach consists of pruning the state space based on the most likely trajectory between states and defining artificial goal states that guide the solution towards the original goals. We also defined a class of short-sighted models that includes depth-based and trajectory-based short-sighted SSPs and proved that SSiPP always terminates and is asymptotically optimal for short-sighted models in this class.

We empirically compared trajectory-based SSiPP with depth-based SSiPP and other state-of-the-art planners in the triangle tireworld. Trajectory-based SSiPP outperforms all the other planners and it is the only planner able to scale up to problem number 60, a problem in which the optimal solution contains approximately $10^{70}$ states, under the IPPC evaluation methodology.

# References

[1] F. W. Trevizan and M. M. Veloso. Short-sighted stochastic shortest path problems. In *In Proc. of the 22nd International Conference on Automated Planning and Scheduling (ICAPS)*, 2012.

[2] D. Bertsekas and J. N. Tsitsiklis. *Neuro-Dynamic Programming*. Athena Scientific, 1996.

[3] A.G. Barto, S.J. Bradtke, and S.P. Singh. Learning to act using real-time dynamic programming. *Artificial Intelligence*, 72(1-2):81–138, 1995.

[4] B. Bonet and H. Geffner. Labeled RTDP: Improving the convergence of real-time dynamic programming. In *Proc. of the 13th International Conference on Automated Planning and Scheduling (ICAPS)*, 2003.

[5] H.B. McMahan, M. Likhachev, and G.J. Gordon. Bounded real-time dynamic programming: RTDP with monotone upper bounds and performance guarantees. In *Proc. of the 22nd International Conference on Machine Learning (ICML)*, 2005.

[6] Trey Smith and Reid G. Simmons. Focused Real-Time Dynamic Programming for MDPs: Squeezing More Out of a Heuristic. In *Proc. of the 21st National Conference on Artificial Intelligence (AAAI)*, 2006.

[7] S. Sanner, R. Goetschalckx, K. Driessens, and G. Shani. Bayesian real-time dynamic programming. In *Proc. of the 21st International Joint Conference on Artificial Intelligence (IJCAI)*, 2009.

[8] S. Yoon, A. Fern, and R. Givan. FF-Replan: A baseline for probabilistic planning. In *Proc. of the 17th International Conference on Automated Planning and Scheduling (ICAPS)*, 2007.

[9] H.L.S. Younes, M.L. Littman, D. Weissman, and J. Asmuth. The first probabilistic track of the international planning competition. *Journal of Artificial Intelligence Research*, 24(1):851–887, 2005.

[10] F. Teichteil-Koenigsbuch, G. Infantes, and U. Kuter. RFF: A robust, FF-based mdp planning algorithm for generating policies with low probability of failure. *3rd International Planning Competition (IPPC-ICAPS)*, 2008.

[11] D. Bryce and O. Buffet. 6th International Planning Competition: Uncertainty Track. In *3rd International Probabilistic Planning Competition (IPPC-ICAPS)*, 2008.

[12] S. Yoon, A. Fern, R. Givan, and S. Kambhampati. Probabilistic planning via determinization in hindsight. In *Proc. of the 23rd National Conference on Artificial Intelligence (AAAI)*, 2008.

[13] S. Yoon, W. Ruml, J. Benton, and M. B. Do. Improving Determinization in Hindsight for Online Probabilistic Planning. In *Proc. of the 20th International Conference on Automated Planning and Scheduling (ICAPS)*, 2010.

[14] I. Little and S. Thiébaux. Probabilistic planning vs replanning. In *Proc. of ICAPS Workshop on IPC: Past, Present and Future*, 2007.

[15] J. Pearl. *Heuristics: Intelligent Search Strategies for Computer Problem Solving*. Addison-Wesley, Menlo Park, California, 1985.

[16] Levente Kocsis and Csaba Szepesvri. Bandit based Monte-Carlo Planning. In *Proc. of the European Conference on Machine Learning (ECML)*, 2006.

[17] T. Dean, L.P. Kaelbling, J. Kirman, and A. Nicholson. Planning under time constraints in stochastic domains. *Artificial Intelligence*, 76(1-2):35–74, 1995.

[18] D.P. Bertsekas and J.N. Tsitsiklis. An analysis of stochastic shortest path problems. *Mathematics of Operations Research*, 16(3):580–595, 1991.

[19] D.P. Bertsekas. *Dynamic Programming and Optimal Control*. Athena Scientific, 1995.

[20] Blai Bonet and Robert Givan. 2th International Probabilistic Planning Competition (IPPC-ICAPS). `http://www.ldc.usb.ve/~bonet/ipc5/` (accessed on Dec 13, 2011), 2007.

